# Localized Sliced Inverse Regression

**Qiang Wu, Sayan Mukherjee**
Department of Statistical Science
Institute for Genome Sciences & Policy
Department of Computer Science
Duke University, Durham
NC 27708-0251, U.S.A
{qiang, sayan}@stat.duke.edu

**Feng Liang**
Department of Statistics
University of Illinois at Urbana-Champaign
IL 61820, U.S.A.
liangf@uiuc.edu

## Abstract

We developed localized sliced inverse regression for supervised dimension reduction. It has the advantages of preventing degeneracy, increasing estimation accuracy, and automatic subclass discovery in classification problems. A semi-supervised version is proposed for the use of unlabeled data. The utility is illustrated on simulated as well as real data sets.

## 1   Introduction

The importance of dimension reduction for predictive modeling and visualization has a long and central role in statistical graphics and computation In the modern context of high-dimensional data analysis this perspective posits that the functional dependence between a response variable $y$ and a large set of explanatory variables $x \in \mathbb{R}^p$ is driven by a low dimensional subspace of the $p$ variables. Characterizing this predictive subspace, supervised dimension reduction, requires both the response and explanatory variables. This problem in the context of linear subspaces or Euclidean geometry has been explored by a variety of statistical models such as sliced inverse regression (SIR, [10]), sliced average variance estimation (SAVE, [3]), principal Hessian directions (pHd, [11]), (conditional) minimum average variance estimation (MAVE, [18]), and extensions to these approaches. To extract nonlinear subspaces, one can apply the aforementioned linear algorithms to the data mapped into a feature space induced by a kernel function [13, 6, 17].

In machine learning community research on nonlinear dimension reduction in the spirit of [19] has been developed of late. This has led to a variety of manifold learning algorithms such as isometric mapping (ISOMAP, [16]), local linear embedding (LLE, [14]), Hessian Eigenmaps [5], and Laplacian Eigenmaps [1]. Two key differences exist between the paradigm explored in this approach and that of supervised dimension reduction. The first difference is that the above methods are unsupervised in that the algorithms take into account only the explanatory variables. This issue can be addressed by extending the unsupervised algorithms to use the label or response data [7]. The bigger problem is that these manifold learning algorithms do not operate on the space of the explanatory variables and hence do not provide a predictive submanifold onto which the data should be projected. These methods are based on embedding the observed data onto a graph and then using spectral properties of the embedded graph for dimension reduction. The key observation in all of these manifold algorithms is that metrics must be local and properties that hold in an ambient Euclidean space are true locally on smooth manifolds.

This suggests that the use of local information in supervised dimension reduction methods may be of use to extend methods for dimension reduction to the setting of nonlinear subspaces and submanifolds of the ambient space. In the context of mixture modeling for classification two approaches have been developed [9, 15].

In this paper we extend SIR by taking into account the local structure of the explanatory variables. This localized variant of SIR, LSIR, can be used for classification as well as regression applications. Though the predictive directions obtained by LSIR are linear ones, they coded nonlinear information. Another advantage of our approach is that ancillary unlabeled data can be easily added to the dimension reduction analysis – semi-supervised learning.

The paper is arranged as follows. LSIR is introduced in Section 2 for continuous and categorical response variables. Extensions are discussed in Section 3. The utility with respect to predictive accuracy as well as exploratory data analysis via visualization is demonstrated on a variety of simulated and real data in Sections 4 and 5. We close with discussions in Section 6.

## 2  Localized SIR

We start with a brief review of SIR method and remark that the failure of SIR in some situations is caused by ignoring local structures. Then we propose a generalization of SIR, called localized SIR, by incorporating some localization idea from manifold learning. Connection to some existing work is addressed at the end.

### 2.1  Sliced inverse regression

Assume the functional dependence between a response variable $Y$ and an explanatory variable $X \in \mathbb{R}^p$ is given by

$$Y = f(\beta_1^t X, \ldots, \beta_L^t X, \epsilon), \tag{1}$$

where $\beta_l$'s are unknown orthogonal vectors in $\mathbb{R}^p$ and $\epsilon$ is noise independent of $X$. Let $B$ denote the $L$-dimensional subspace spanned by $\beta_l$'s. Then $P_B X$, where $P_B$ denotes the projection operator onto space $B$, provides a sufficient summary of the information in $X$ relevant to $Y$. Estimating $B$ or $\beta_l$'s becomes the central problem in supervised dimension reduction. Though we define $B$ here via a heuristic model assumption (1), a general definition based on conditional independence between $Y$ and $X$ given $P_B X$ can be found in [4]. Following [4], we refer to $B$ as the dimension reduction (d.r.) subspace and $\beta_l$'s the d.r. directions.

Slice inverse regression (SIR) model is introduced [10] to estimate the d.r. directions. The idea underlying this approach is that, if $X$ has an identity covariance matrix, the centered inverse regression curve $\mathbb{E}(X|Y) - \mathbb{E}X$ is contained in the d.r. space $B$ under some design conditions; see [10] for details. According to this result the d.r. directions $\beta_l$'s are given by the top eigenvectors of the covariance matrix $\Gamma = \mathrm{Cov}(\mathbb{E}(X|Y))$. In general when the covariance matrix of $X$ is $\Sigma$, the $\beta_l$'s can be obtained by solving a generalized eigen decomposition problem

$$\Gamma\beta = \lambda\Sigma\beta.$$

A simple SIR algorithm operates as the following on a set of samples $\{(x_i, y_i)\}_{i=1}^n$:

1. Compute an empirical estimate of $\Sigma$,

$$\hat{\Sigma} = \frac{1}{n}\sum_{i=1}^n (x_i - m)(x_i - m)^T$$

   where $m = \frac{1}{n}\sum_{i=1}^n x_i$ is the sample mean.

2. Divide the samples into $H$ groups (or *slices*) $G_1, \ldots, G_H$ according to the value of $y$. Compute an empirical estimate of $\Gamma$,

$$\hat{\Gamma} = \sum_{i=1}^H \frac{n_h}{n}(m_h - m)(m_h - m)^T$$

   where $m_h = \frac{1}{n_h}\sum_{j \in G_h} x_j$ is the sample mean for group $G_h$ with $n_h$ being the group size.

3. Estimate the d.r. directions $\beta$ by solving a generalized eigen problem

$$\hat{\Gamma}\beta = \lambda\hat{\Sigma}\beta. \tag{2}$$

When $Y$ takes categorical values as in classification problems, it is natural to divide the data into different groups by their group labels. Then SIR is equivalent to Fisher discriminant analysis (FDA).[1]

Though SIR has been widely used for dimension reduction and yielded many useful results in practice, it has some known problems. For example, it is easy to construct a function $f$ such that $\mathbb{E}(X|Y = y) = 0$ then SIR fails to retrieve any useful directions [3]. The degeneracy of SIR has also restricted its use in binary classification problems where only one direction can be obtained. The failure of SIR in these scenario is partly because the algorithm uses just the mean, $\mathbb{E}(X|Y = y)$, as a summary of the information in each slice, which apparently is not enough. Generalizations of SIR include SAVE [3], SIR-II [12] and covariance inverse regression estimation (CIRE, [2]) that exploit the information from the second moment of the conditional distribution of $X|Y$. However in some scenario the information in each slice can not be well described by a global statistics. For example, similar to the *multimodal* situation considered by [15], the data in a slice may form two clusters, then a good description of the data would not be a single number such as any moments, but the two cluster centers. Next we will propose a new algorithm that is a generalization of SIR based on local structures of $X$ in each slice.

## 2.2 Localization

A key principle in manifold learning is that the Euclidean representation of a data point in $\mathbb{R}^p$ is only meaningful locally. Under this principle, it is dangerous to calculate the slice average $m_h$, whenever the slice contains data that are far away. Instead some kind of local averages should be considered. Motivated by this idea we introduce a localized SIR (LSIR) method for dimension reduction.

Here is the intuition for LSIR. Let us start with the transformed data set where the empirical covariance is identity, for example, the data set after PCA. In the original SIR method, we shift every data point $x_i$ to the corresponding group average, then apply PCA on the new data set to identify SIR directions. The underline rational for this approach is that if a direction does not differentiate different groups well, the group means projected to that direction would be very close, therefore the variance of the new data set will be small at that direction. A natural way to incorporate localization idea into this approach is to shift each data point $x_i$ to the average of a local neighborhood instead of the average of its global neighborhood (i.e., the whole group). In manifolds learning, local neighborhood is often chosen by $k$ nearest neighborhood ($k$-NN). Different from manifolds learning that is designed for unsupervised learning, the neighborhood selection for LSIR that is designed for supervised learning will also incorporate information from the response variable $y$.

Here is the mathematical description of LSIR. Recall that the group average $m_h$ is used in estimating $\Gamma = \text{Cov}(\mathbb{E}(X|Y))$. The estimate $\hat{\Gamma}$ is equivalent to the sample covariance of a data set $\{m_i\}_{i=1}^n$ where $m_i = m_h$, average of the group $G_h$ to which $x_i$ belongs. In our LSIR algorithm, we set $m_i$ equal to some local average, and then use the corresponding sample covariance matrix to replace $\hat{\Gamma}$ in equation (2). Below we give the details of our LSIR algorithm:

1. Compute $\hat{\Sigma}$ as in SIR.
2. Divide the samples into $H$ groups as in SIR. For each sample $(x_i, y_i)$ we compute

$$m_{i,loc} = \frac{1}{k} \sum_{j \in s_i} x_j,$$

   where, with $h$ being the group so that $i \in G_h$,

$$s_i = \{j : x_j \text{ belongs to the } k \text{ nearest neighbors of } x_i \text{ in } G_h\}.$$

   Then we compute a localized version of $\Gamma$ by

$$\hat{\Gamma}_{loc} = \frac{1}{n} \sum_{i=1}^n (m_{i,loc} - m)(m_{i,loc} - m)^T.$$

3. Solve the generalized eigen decomposition problem

$$\hat{\Gamma}_{loc} \beta = \lambda \hat{\Sigma} \beta. \tag{3}$$

The neighborhood size $k$ in LSIR is a tuning parameter specified by users. When $k$ is large enough, say, larger than the size of any group, then $\hat{\Gamma}_{loc}$ is the same as $\hat{\Gamma}$ and LSIR recovers all SIR directions. With a moderate choice of $k$, LSIR uses the local information within each slice and is expected to retrieve directions lost by SIR in case of SIR fails due to degeneracy.

For classification problems LSIR becomes a localized version of FDA. Suppose the number of classes is $C$, then the estimate $\hat{\Gamma}$ from the original FDA is of rank at most $C - 1$, which means FDA can only estimate at most $C - 1$ directions. This is why FDA is seldom used for binary classification problems where $C = 2$. In LSIR we use more points to describe the data in each class. Mathematically this is reflected by the increase of the rank of $\hat{\Gamma}_{loc}$ that is no longer bounded by $C$ and hence produces more directions. Moreover, if for some classes the data is composed of several sub-clusters, LSIR can automatically identify these sub-cluster structures. As showed in one of our examples, this property of LSIR is very useful in data analysis such as cancer subtype discovery using genomic data.

### 2.3   Connection to Existing Work

The idea of localization has been introduced to dimension reduction for classification problems before. For example, the local discriminant information (LDI) introduced by [9] is one of the early work in this area. In LDI, the local information is used to compute the between-group covariance matrix $\Gamma_i$ over a nearest neighborhood at every data point $x_i$ and then estimate the d.r. directions by the top eigenvector of the averaged between-group matrix $\frac{1}{n} \sum_{i=1}^{n} \Gamma_i$. The local Fisher discriminant analysis (LFDA) introduced by [15] can be regarded as an improvement of LDI with the within-class covariance matrix also being localized.

Comparing to these two approaches, LSIR utilizes the local information directly at the point level. One advantage of this simple localization is computation. For example, for a problem of $C$ classes, LDI needs to compute $nC$ local mean points and $n$ between-group covariance matrices, while LSIR computes only $n$ local mean points and one covariance matrix. Another advantage is LSIR can be easily extended to handle unlabeled data in semi-supervised learning as explained in the next section. Such an extension is less straightforward for the other two approaches that operate on the covariance matrices instead of data points.

## 3   Extensions

**Regularization.** When the matrix $\hat{\Sigma}$ is singular or has a very large condition number, which is common in high-dimensional problems, the generalized eigen-decomposition problems (3) is unstable. Regularization techniques are often introduced to address this issue [20]. For LSIR we adopt the following regularization:

$$\hat{\Gamma}_{loc}\beta = \lambda(\hat{\Sigma} + s)\beta \tag{4}$$

where the regularization parameter $s$ can be chosen by cross validation or other criteria (e.g. [20]).

**Semi-supervised learning.** In semi-supervised learning some data have $y$'s (*labeled* data) and some do not (*unlabeled* data). How to incorporate the information from unlabeled data has been the main focus of research in semi-supervised learning. Our LSIR algorithm can be easily modified to take the unlabeled data into consideration. Since $y$ of an unlabeled sample can take any possible values, we put the unlabeled data into every slice. So the neighborhood $s_i$ is defined as the following: for any point in the $k$-NN of $x_i$, it belongs to $s_i$ if it is unlabeled, or if it is labeled and belongs to the same slice as $x_i$.

## 4   Simulations

In this section we apply LSIR to several synthetic data sets to illustrate the power of LSIR. The performance of LSIR is compared with other dimension reduction methods including SIR, SAVE, pHd, and LFDA.

| Method | SAVE | pHd | LSIR ($k = 20$) | LSIR ($k = 40$) |
|--------|------|-----|-----------------|-----------------|
| Accuracy | $0.3451(\pm 0.1970)$ | $0.3454(\pm 0.1970)$ | $0.9534(\pm.0004)$ | $0.9011(\pm.0008)$ |

Table 1: Estimation accuracy (and standard deviation) of various dimension reduction methods for semisupervised learning in Example 1.

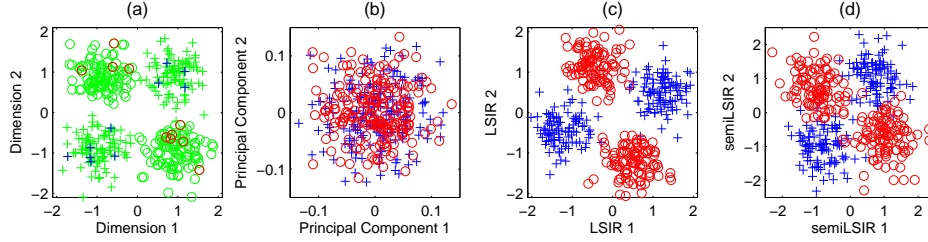

Figure 1: Result for Example 1. (a) Plot of data in the first two dimensions, where '+' corresponds to $y = 1$ while 'o' corresponds to $y = -1$. The data points in red and blue are labeled and the ones in green are unlabeled when the semisupervised setting is considered. (b) Projection of data to the first two PCA directions. (c) Projection of data to the first two LSIR directions when all the $n = 400$ data points are labeled. (d) Projection of the data to the first two LSIR directions when only 20 points as indicated in (a) are labeled.

Let $\hat{B} = (\hat{\beta}_1, \cdots, \hat{\beta}_L)$ denote an estimate of the d.r. subspace $B$ where its columns $\hat{\beta}_l$'s are the estimated d.r. directions. We introduce the following metric to measure the accuracy:

$$\text{Accuracy}(\hat{B}, B) = \frac{1}{L} \sum_{i=1}^{L} \|P_B \hat{\beta}_i\|^2 = \frac{1}{L} \sum_{i=1}^{L} \|(BB^T) \hat{\beta}_i\|^2.$$

In LSIR the influence of the parameter $k$, the size of local neighborhoods, is subtle. In our simulation study, we found it usually good enough to choose $k$ between 10 to 20, except for the semisupervised setting (e.g. Example 1 below). But further study and a theoretical justification are necessary.

*Example* 1. Consider a binary classification problem on $\mathbb{R}^{10}$ where the d.r. directions are the first two dimensions and the remaining eight dimensions are Gaussian noise. The data in the first two relevant dimensions are plotted in Figure 1(a) with sample size $n = 400$. For this example SIR cannot identify the two d.r. directions because the group averages of the two groups are roughly the same for the first two dimensions, due to the symmetry in the data. Using local average instead of group average, LSIR can find both directions, see Figure 1(c). But so do SAVE and pHd since the high-order moments also behave differently in the two groups.

Next we create a data set for semi-supervised learning by randomly selecting 20 samples, 10 from each group, to be labeled and setting others to be unlabeled. The directions from PCA where one ignores the labels do not agree with the discriminant directions as shown in Figure 1(b). So to retrieve the relevant directions, the information from the labeled points has to be taken consideration. We evaluate the accuracy of LSIR (the semi-supervised version), SAVE and pHd where the latter two are operated on just the labeled set. We repeat this experiment 20 times and each time select a different random set to be labeled. The averaged accuracy is reported in Table 1. The result for one iteration is displayed in Figure 1 where the labeled points are indicated in (a) and the projection to the top two directions from LSIR (with $k = 40$) is in (d). All the results clearly indicate that LSIR out-performs the other two supervised dimension reduction methods.

*Example* 2. We first generate a 10-dimensional data set where the first three dimensions are the Swiss roll data [14]:

$$X_1 = t \cos t, \quad X_2 = 21h, \quad X_3 = t \sin t,$$

where $t = \frac{3\pi}{2}(1 + 2\theta)$, $\theta \sim \text{Uniform}(0, 1)$ and $h \sim \text{Uniform}([0, 1])$. The remaining 7 dimensions are independent Gaussian noises. Then all dimensions are normalized to have unit variance. Consider the following function:

$$Y = \sin(5\pi\theta) + h^2 + \epsilon, \qquad \epsilon \sim \text{N}(0, 0.1^2). \tag{5}$$

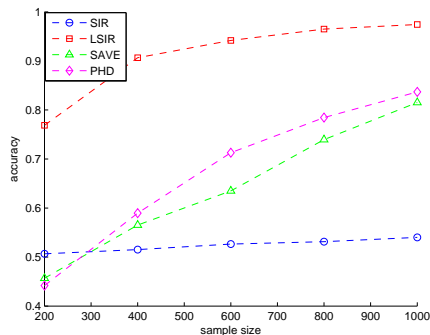

Figure 2: Estimation accuracy of various dimension methods for example 2.

We randomly choose $n$ samples as a training set and let $n$ change from 200 to 1000 and compare the estimation accuracy for LSIR with SIR, SAVE and pHd. The result is showed in Figure 2. SAVE and pHd outperform SIR, but are still much worse comparing to LSIR.

Note that Swiss roll (the first three dimensions) is a benchmark data set in manifolds learning, where the goal is to "unroll" the data into the intrinsic two dimensional space. Since LSIR is a linear dimension reduction method we do not expect LSIR to unroll the data, but expect to retrieve the dimensions relevant to the prediction of $Y$. Meanwhile, with the noise, manifolds learning algorithms will not unroll the data either since the dominant directions are now the noise dimensions.

*Example* 3. (Tai Chi) The Tai Chi figure is well known in Asian culture where the concepts of Yin-Yang provide the intellectual framework for much of ancient Chinese scientific development. A 6-dimensional data set for this example is generated as follows: $X_1$ and $X_2$ are from the Tai Chi structure as shown in Figure 3(a) where the Yin and Yang regions are assigned class labels $Y = -1$ and $Y = 1$ respectively. $X_3, \ldots, X_6$ are independent random noise generated by $N(0,1)$.

The Tai Chi data set was first used as a dimension reduction example in [12, Chapter 14]. The correct d.r. subspace $B$ is span($\mathbf{e}_1, \mathbf{e}_2$). SIR, SAVE and pHd are all known to fail for this example. By taking the local structure into account, LSIR can easily retrieve the relevant directions. Following [12], we generate $n = 1000$ samples as the training data, then run LSIR with $k = 10$ and repeat 100 times. The average accuracy is $98.6\%$ and the result from one run is shown in Figure 3. For comparison we also applied LFDA for this example. The average accuracy is $82\%$ which is much better than SIR, SAVE and pHd but worse than LSIR.

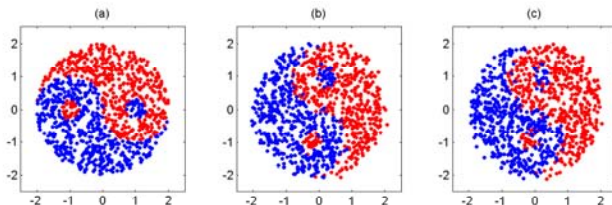

Figure 3: Result for Tai Chi example. (a) The training data in first two dimensions; (b) The training data projected onto the first two LSIR directions; (c) An independent test data projected onto the first two LSIR directions.

## 5 Applications

In this section we apply our LSIR methods to two real data sets.

### 5.1 Digits recognition

The MNIST data set (Y. LeCun, *http://yann.lecun.com/exdb/mnist/*) is a well known benchmark data set for classification learning. It contains $60,000$ images of handwritten digits as training data and $10,000$ images as test data. This data set is commonly believed to have strong nonlinear structures.

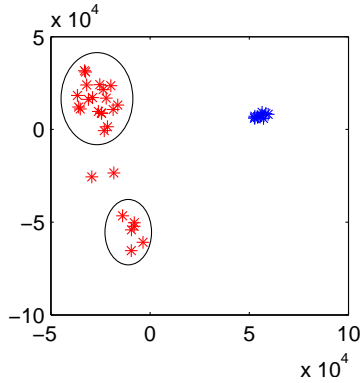

Figure 4: Result for leukemia data by LSIR. Red points are ALL and blue ones are AML

In our simulations, we randomly sampled $1000$ images ($100$ samples for each digit) as training set. We apply LSIR and computed $d = 20$ e.d.r. directions. Then we project the training data and $10000$ test data onto these directions. Using a k-nearest neighbor classifier with $k = 5$ to classify the test data, we report the classification error over $100$ iterations in Table 2. Compared with SIR method, the classification accuracy are increased for almost all digits. The improvement for digits $2, 3, 5$ is much significant.

| digits | 0 | 1 | 2 | 3 | 4 | 5 | 6 | 7 | 8 | 9 | average |
|--------|--------|--------|--------|--------|--------|--------|--------|--------|--------|--------|---------|
| LSIR | 0.0350 | 0.0098 | 0.1363 | 0.1055 | 0.1309 | 0.1175 | 0.0445 | 0.1106 | 0.1417 | 0.1061 | 0.0927 |
| SIR | 0.0487 | 0.0292 | 0.1921 | 0.1723 | 0.1327 | 0.2146 | 0.0816 | 0.1354 | 0.1981 | 0.1533 | 0.1358 |

Table 2: Classification error rate for digits classification by SIR and LSIR.

## 5.2 Gene expression data

Cancer classification and discovery using gene expression data becomes an important technique in modern biology and medical science. In gene expression data number of genes is huge (usually up to thousands) and the samples is quite limited. As a typical large $p$ small $n$ problem, dimension reduction plays very essential role to understand the data structure and make inference.

*Leukemia classification*. We consider leukemia classification in [8]. This data has 38 training samples and 34 test samples. The training sample has two classes, AML and ALL, and the class ALL has two subtypes. We apply SIR and LSIR to this data. The classification accuracy is similar by predicting the test data with 0 or 1 error. An interesting point is that LSIR automatically realizes subtype discovery while SIR cannot. By project the training data onto the first two directions (Figure 4), we immediately notice that the ALL has two subtypes. It turns out that the 6-samples cluster are T-cell ALL and the 19-samples cluster is B-cell ALL samples. Note that there are two samples (which are T-cell ALL) cannot be assigned to each subtype only by visualization. This means LSIR only provides useful subclass knowledge for future research but itself may not a perfect clustering method.

## 6 Discussion

We developed LSIR method for dimension reduction by incorporating local information into the original SIR. It can prevent degeneracy, increase estimation accuracy, and automatically identify subcluster structures. Regularization technique is introduced for computational stability. A semi-supervised version is developed for the use of unlabeled data. The utility is illustrated on synthetic as well as real data sets.

Since LSIR involves only linear operations on the data points, it is straightforward to extend it to kernel models [17] via the so-called kernel trick. An extension of LSIR along this direction

can be helpful to realize nonlinear dimension reduction directions and to reduce the computational complexity in case of $p \gg n$.

Further research on LSIR and its kernelized version includes their asymptotic properties such as consistency and statistically more rigorous approaches for the choice of $k$, the size of local neighborhoods, and $L$, the dimensionality of the reduced space.

## Footnotes

[1] FDA is referred to as linear discriminant analysis (LDA) in some literatures.

## References

[1] M. Belkin and P. Niyogi. Laplacian eigenmaps for dimensionality reduction and data representation. *Neural Computation*, 15(6):1373–1396, 2003.

[2] R. Cook and L. Ni. Using intra-slice covariances for improved estimation of the central subspace in regression. *Biometrika*, 93(1):65–74, 2006.

[3] R. Cook and S. Weisberg. Disussion of li (1991). *J. Amer. Statist. Assoc.*, 86:328–332, 1991.

[4] R. Cook and X. Yin. Dimension reduction and visualization in discriminant analysis (with discussion). *Aust. N. Z. J. Stat.*, 43(2):147–199, 2001.

[5] D. Donoho and C. Grimes. Hessian eigenmaps: new locally linear embedding techniques for highdimensional data. *PNAS*, 100:5591–5596, 2003.

[6] K. Fukumizu, F. R. Bach, and M. I. Jordan. Kernel dimension reduction in regression. Annals of Statistics, to appear, 2008.

[7] A. Globerson and S. Roweis. Metric learning by collapsing classes. In Y. Weiss, B. Schölkopf, and J. Platt, editors, *Advances in Neural Information Processing Systems 18*, pages 451–458. MIT Press, Cambridge, MA, 2006.

[8] T. Golub, D. Slonim, P. Tamayo, C. Huard, M. Gaasenbeek, J. Mesirov, H. Coller, M. Loh, J. Downing, M. Caligiuri, C. Bloomfield, and E. Lander. Molecular classification of cancer: class discovery and class prediction by gene expression monitoring. *Science*, 286:531–537, 1999.

[9] T. Hastie and R. Tibshirani. Discrminant adaptive nearest neighbor classification. *IEEE Transacations on Pattern Analysis and Machine Intelligence*, 18(6):607–616, 1996.

[10] K. Li. Sliced inverse regression for dimension reduction (with discussion). *J. Amer. Statist. Assoc.*, 86:316–342, 1991.

[11] K. C. Li. On principal hessian directions for data visulization and dimension reduction: another application of stein's lemma. *J. Amer. Statist. Assoc.*, 87:1025–1039, 1992.

[12] K. C. Li. High dimensional data analysis via the sir/phd approach, 2000.

[13] J. Nilsson, F. Sha, and M. I. Jordan. Regression on manifold using kernel dimension reduction. In *Proc. of ICML 2007*, 2007.

[14] S. Roweis and L. Saul. Nonlinear dimensionality reduction by locally linear embedding. *Science*, 290:2323–2326, 2000.

[15] M. Sugiyam. Dimension reduction of multimodal labeled data by local fisher discriminatn analysis. *Journal of Machine Learning Research*, 8:1027–1061, 2007.

[16] J. Tenenbaum, V. de Silva, and J. Langford. A global geometric framework for nonlinear dimensionality reduction. *Science*, 290:2319–2323, 2000.

[17] Q. Wu, F. Liang, and S. Mukherjee. Regularized sliced inverse regression for kernel models. Technical report, ISDS Discussion Paper, Duke University, 2007.

[18] Y. Xia, H. Tong, W. Li, and L.-X. Zhu. An adaptive estimation of dimension reduction space. *J. R. Statist. Soc. B*, 64(3):363–410, 2002.

[19] G. Young. Maximum likelihood estimation and factor analysis. *Psychometrika*, 6:49–53, 1941.

[20] W. Zhong, P. Zeng, P. Ma, J. S. Liu, and Y. Zhu. RSIR: regularized sliced inverse regression for motif discovery. *Bioinformatics*, 21(22):4169–4175, 2005.

